# Scheduling Straight-Line Code Using Reinforcement Learning and Rollouts

**Amy McGovern and Eliot Moss**
{*amy*|*moss@cs.umass.edu*}
Department of Computer Science
University of Massachusetts, Amherst
Amherst, MA 01003

## Abstract

The execution order of a block of computer instructions can make a difference in its running time by a factor of two or more. In order to achieve the best possible speed, compilers use heuristic schedulers appropriate to each specific architecture implementation. However, these heuristic schedulers are time-consuming and expensive to build. In this paper, we present results using both rollouts and reinforcement learning to construct heuristics for scheduling basic blocks. The rollout scheduler outperformed a commercial scheduler, and the reinforcement learning scheduler performed almost as well as the commercial scheduler.

## 1 Introduction

Although high-level code is generally written as if it were going to be executed sequentially, many modern computers are pipelined and allow for the simultaneous issue of multiple instructions. In order to take advantage of this feature, a scheduler needs to reorder the instructions in a way that preserves the semantics of the original high-level code while executing it as quickly as possible. An efficient schedule can produce a speedup in execution of a factor of two or more. However, building a scheduler can be an arduous process. Architects developing a new computer must manually develop a specialized instruction scheduler each time a change is made in the proposed system. Building a scheduler automatically can save time and money. It can allow the architects to explore the design space more thoroughly and to use more accurate metrics in evaluating designs.

Moss et al. (1997) showed that supervised learning techniques can induce excellent basic block instruction schedulers for the Digital Alpha 21064 processor. Although all of the supervised learning methods performed quite well, they shared several limitations. Supervised learning requires exact input/output pairs. Generating these training pairs requires an optimal scheduler that searches every valid permutation of the instructions within a basic block and saves the optimal permutation (the schedule with the smallest running time). However, this search was too time-consuming to perform on blocks with more than 10 in-

structions, because optimal instruction scheduling is NP-hard. Using a semi-supervised method such as reinforcement learning or rollouts does not require generating training pairs, so the method can be applied to larger basic blocks and can be trained without knowing optimal schedules.

## 2  Domain Overview

Moss et al. (1997) gave a full description of the domain. This study presents an overview, necessary details, our experimental method and detailed results for both rollouts and reinforcement learning.

We focused on scheduling *basic blocks* of instructions on the 21064 version (DEC, 1992) of the Digital Alpha processor (Sites, 1992). A basic block is a set of instructions with a single entry point and a single exit point. Our schedulers could reorder instructions within a basic block but could not rewrite, add, or remove any instructions. The goal of each scheduler is to find a least-cost valid ordering of the instructions. The cost is defined as the simulated execution time of the block. A valid ordering is one that preserves the semantically necessary ordering constraints of the original code. We insure validity by creating a dependency graph that directly represents those necessary ordering relationships. This graph is a directed acyclic graph (DAG).

The Alpha 21064 is a dual-issue machine with two different execution pipelines. Dual issue occurs only if a number of detailed conditions hold, e.g., the two instructions match the two pipelines. An instruction can take anywhere from one to many tens of cycles to execute. Researchers at Digital have a publicly available 21064 simulator that also includes a heuristic scheduler for basic blocks. We call that scheduler *DEC*. The simulator gives the running time for a given scheduled block assuming all memory references hit the cache and all resources are available at the beginning of the block. All of our schedulers used a greedy algorithm to schedule the instructions, i.e., they built schedules sequentially from beginning to end with no backtracking.

In order to test each scheduling algorithm, we used the 18 SPEC95 benchmark programs. Ten of these programs are written in FORTRAN and contain mostly floating point calculations. Eight of the programs are written in C and focus more on integer, string, and pointer calculations. Each program was compiled using the commercial Digital compiler at the highest level of optimization. We call the schedules output by the compiler *ORIG*. This collection has 447,127 basic blocks, containing 2,205,466 instructions.

## 3  Rollouts

Rollouts are a form of Monte Carlo search, first introduced by Tesauro and Galperin (1996) for use in backgammon. Bertsekas et al. (1997a,b) have explored rollouts in other domains and proven important theoretical results. In the instruction scheduling domain, rollouts work as follows: suppose the scheduler comes to a point where it has a partial schedule and a set of (more than one) candidate instructions to add to the schedule. For each candidate, the scheduler appends it to the partial schedule and then follows a fixed policy $\pi$ to schedule the remaining instructions. When the schedule is complete, the scheduler evaluates the running time and returns. When $\pi$ is stochastic, this rollout can be repeated many times for each instruction to achieve a measure of the average expected outcome. After rolling out each candidate, the scheduler picks the one with the best average running time.

Our first set of rollout experiments compared three different rollout policies $\pi$. The theory developed by Bertsekas et al. (1997a,b) proved that if we used the DEC scheduler as $\pi$, we would perform no worse than DEC. An architect proposing a new machine might not have a good heuristic available to use as $\pi$, so we also considered policies more likely to be available. The first was the random policy, *RANDOM-$\pi$*, which is a choice that is clearly always available. Under this policy, the rollout makes all choices randomly. We also used

the ordering produced by the optimizing compiler ORIG, denoted *ORIG-$\pi$*. The last rollout policy tested was the DEC scheduler itself, denoted *DEC-$\pi$*.

The scheduler performed only one rollout per candidate instruction when using ORIG-$\pi$ and DEC-$\pi$ because they are deterministic. We used 25 rollouts for RANDOM-$\pi$. After performing a number of rollouts for each candidate instruction, we chose the instruction with the best average running time. As a baseline scheduler, we also scheduled each block randomly. Because the running time increases quadratically with the number of rollouts, we focused our rollout experiments on one program in the SPEC95 suite: *applu*.

Table 1 gives the performance of each rollout scheduler as compared to the DEC scheduler on all 33,007 basic blocks of size 200 or less from applu. To assess the performance of each rollout policy $\pi$, we used the ratio of the weighted execution time of the rollout scheduler to the weighted execution time of the DEC scheduler. More concisely, the performance measure was:

$$\text{ratio} = \frac{\sum_{\text{all blocks}} \text{rollout scheduler execution time} * \text{number of times block is executed}}{\sum_{\text{all blocks}} \text{DEC scheduler execution time} * \text{number of times block is executed}}$$

This means that a faster running time on the part of our scheduler would give a smaller ratio.

| Scheduler | Ratio | Scheduler | Ratio |
|-----------|-------|-----------|-------|
| Random | 1.3150 | RANDOM-$\pi$ | 1.0560 |
| ORIG-$\pi$ | 0.9895 | DEC-$\pi$ | 0.9875 |

Table 1: Ratios of the weighted execution time of the rollout scheduler to the DEC scheduler. A ratio of less than one means that the rollouts outperformed the DEC scheduler.

All of the rollout schedulers far outperformed the random scheduler which was 31% slower than DEC. By only adding rollouts, RANDOM-$\pi$ was able to achieve a running time only 5% slower than DEC. Only the schedulers using ORIG-$\pi$ and DEC-$\pi$ as a model outperformed the DEC scheduler. Using ORIG-$\pi$ and DEC-$\pi$ for rollouts produced a schedule that was 1.1% faster than the DEC scheduler on average. Although this improvement may seem small, the DEC scheduler is known to make optimal choices 99.13% of the time for blocks of size 10 or less (Stefanović, 1997).

Rollouts were tested only on applu rather than on the entire SPEC95 benchmark suite due to the lengthy computation time. Rollouts are costly because performing $m$ rollouts on $n$ instructions is $O(n^2 m)$, whereas a greedy scheduling algorithm is $O(n)$. Again, because of the time required, we only performed five runs of RANDOM-$\pi$. Since DEC-$\pi$ and ORIG-$\pi$ are deterministic, only one run was necessary. We also ran the random scheduler 5 times. Each number reported above is the geometric mean of the ratios across the five runs.

Part of the motivation behind using rollouts in a scheduler is to obtain fast schedules without spending the time to build a precise heuristic. With this in mind, we explored RANDOM-$\pi$ more closely in a follow-up experiment.

**Evaluation of the number of rollouts**

This experiment considered how performance varies with the number of rollouts. We tested 1, 5, 10, 25, and 50 rollouts per candidate instruction. We also varied the metric for choosing among candidates. Instead of always choosing the instruction with the best average performance, we also experimented with selecting the instruction with the absolute best running time among its rollouts. We hypothesized that selection of the absolute best path might lead to better performance overall. These experiments were performed on all 33,007 basic blocks of size 200 or less from applu.

Figure 1 shows the performance of the rollout scheduler as a function of the number of rollouts. Performance is assessed in the same way as before: ratio of weighted execution

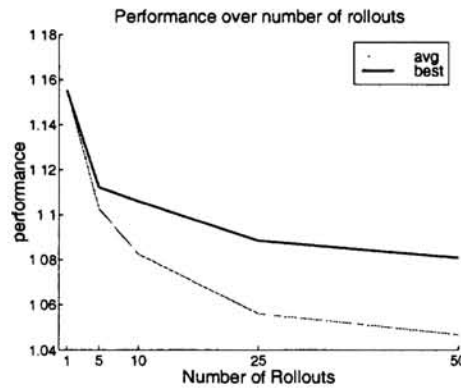

Figure 1: Performance of rollout scheduler with the random model as a function of the number of rollouts and the choice of evaluation function.

times. Thus, a lower number is better. Each data point represents the geometric mean over five different runs. The difference in performance between one rollout and five rollouts using the average choice for each rollout is 1.16 versus 1.10. However, the difference between 25 rollouts and 50 rollouts is only 1.06 versus 1.05. This indicates the tradeoff between schedule quality and the number of rollouts. Also, choosing the instructions with the best rollout schedule did not yield better performance over any numbers of rollouts. We hypothesize that this is due to the stochastic nature of the rollouts. Once the scheduler chooses an instruction, it repeats the rollout process again. By choosing the instruction with the absolute best rollout, there is no guarantee that the scheduler will find that permutation of instructions again on the next rollout. When it chooses the instruction with the best average rollout, the scheduler has a better chance of finding a good schedule on the next rollout.

Although the rollout schedulers performed quite well, the extremely long scheduling time is a major drawback. Using 25 rollouts per block took over 6 hours to schedule one program. Unless this aspect can be improved, rollouts cannot be used for all blocks in a commercial scheduler or in evaluating more than a few proposed machine architectures. However, because rollout scheduling performance is high, rollouts could be used to optimize the schedules on important (long running times or frequently executed) blocks within a program.

## 4 Reinforcement Learning Results

### 4.1 Overview

Reinforcement learning (RL) is a collection of methods for discovering near-optimal solutions to stochastic sequential decision problems (Sutton & Barto, 1998). A reinforcement learning system does not require a teacher to specify correct actions. Instead, the learning agent tries different actions and observes their consequences to determine which actions are best. More specifically, in the reinforcement learning framework, a learning *agent* interacts with an *environment* over a series of discrete time steps $t = 0, 1, 2, 3, \ldots$. At each time $t$, the agent is in some *state*, denoted $s_t$, and chooses an action, denoted $a_t$, which causes the environment to transition to state $s_{t+1}$ and to emit a reward, denoted $r_{t+1}$. The next state and reward depend only on the preceding state and action, but they may depend on it in a stochastic fashion. The objective is to learn a (possibly stochastic) mapping from states to actions called a *policy*, which maximizes the cumulative discounted reward received by the agent. More precisely, the objective is to choose action $a_t$ so as to maximize the expected *return*, $E \left\{ \sum_{i=0}^{\infty} \gamma^i r_{t+i+1} \right\}$, where $\gamma \in [0, 1)$ is a discount-rate parameter.

A common solution strategy is to approximate the *optimal value function* $V^*$, which maps states to the maximal expected return that can be obtained starting in each state and taking the best action. In this paper we use *temporal difference (TD) learning* (Sutton, 1988). In this method, the approximation to $V^*$ is represented by a table with an entry $V(s)$ for every state. After each transition from state $s_t$ to state $s_{t+1}$, under an action with reward $r_{t+1}$, the estimated value function $V(s_t)$ is updated by:

$$V(s_t) \leftarrow V(s_t) + \alpha\left[r_{t+1} + \gamma V(s_{t+1}) - V(s_t)\right]$$

where $\alpha$ is a positive step-size parameter.

## 4.2  Experimental Results

Scheeff et al. (1997) have previously experimented with reinforcement learning in this domain. However, the results were not as good as hoped. Finding the right reward structure was the difficult part of using RL in this domain. Rewarding based on number of cycles to execute the block does not work well as it punishes the learner on long blocks. To normalize for this effect, Scheeff et al. (1997) rewarded based on the cycles per instruction (CPI). However, learning with this reward also did not work well as some blocks have more unavoidable idle time than others. A reward based solely on CPI does not account for this aspect. To account for this variation across blocks, we gave the RL scheduler a final reward of:

$$r = \text{time to execute block} - \max\left(\text{minimum weighted critical path}, \left(\frac{\#\text{ of instructions}}{2}\right)\right)$$

The scheduler received a reward of zero unless the schedule was complete. As the 21064 processor can only issue two instructions at a time, the number of instructions divided by 2 gives an absolute lower bound on the running time. The weighted critical path (wcp) helps to solve the problem of the same size blocks being easier or harder to schedule than others. When a block is harder to execute than another block of the same size, the wcp tends to be higher, thus causing the learner to get a different reward. The wcp is correlated with the predicted number of execution cycles for the DEC scheduler ($r = 0.9$) and the number of instructions divided by 2 is also correlated ($r = 0.78$) with the DEC scheduler. Future experiments will use a weighted combination of these two features to compute the reward.

As with the supervised learning results presented in Moss et al. (1997), the RL system learned a preferential value function between candidate instructions. That is, instead of learning the value of instruction A or instruction B, RL learned the value of choosing instruction A over instruction B. The state space consisted of a tuple of features from a current partial schedule and the two candidate instructions. These features were derived from knowledge of the DEC simulator. The features and our intuition for their importance are summarized in Table 2.

Previous experiments (Moss et al. 1997) showed that the actual value of wcp and e did not matter as much as their relative values. Thus, for those features we used the signum ($\sigma$) of the difference of their values for the two candidate instruction. Signum returns $-1, 0,$ or $1$ depending on whether the value is less than, equal to, or greater than zero. Using this representation, the RL state space consisted of the following tuple, given candidate instruction $x$ and $y$ and partial schedule $p$:

$$\text{state\_vec}(p, x, y) = \langle \text{odd}(p), \text{ic}(x), \text{ic}(y), \text{d}(x), \text{d}(y), \sigma(\text{wcp}(x) - \text{wcp}(y)), \sigma(\text{e}(x) - \text{e}(y)) \rangle$$

This yields 28,800 unique states. Figure 2 shows an example partial schedule, a set of candidate instructions, and the resulting states for the RL system.

The RL scheduler does not learn over states where there are no choices to be made. The last choice point in a trajectory is given the final reward even if further instructions are scheduled from that point. The values of multiple states are updated at each time step because the instruction that is chosen affects the preference function of multiple states. For

| Heuristic Name | Heuristic Description | Intuition for Use |
|---|---|---|
| Odd Partial (odd) | Is the current number of instructions scheduled odd or even? | If TRUE, we're interested in scheduling instructions that can dual-issue with the previous instruction. |
| Instruction Class (ic) | The Alpha's instructions can be divided into equivalence classes with respect to timing properties. | The instructions in each class can be executed only in certain execution pipelines, etc. |
| Weighted Critical Path (wcp) | The height of the instruction in the DAG (the length of the longest chain of instructions dependent on this one), with edges weighted by expected latency of the result produced by the instruction | Instructions on longer critical paths should be scheduled first, since they affect the lower bound of the schedule cost. |
| Actual Dual (d) | Can the instruction dual-issue with the previous scheduled instruction? | If Odd Partial is TRUE, it is important that we find an instruction, if there is one, that can issue in the same cycle with the previous scheduled instruction. |
| Max Delay (e) | The earliest cycle when the instruction can begin to execute, relative to the current cycle; this takes into account any wait for inputs for functional units to become available | We want to schedule instructions that will have their data and functional unit available earliest. |

Table 2: Features for Instructions and Partial Schedule

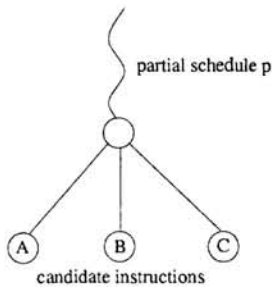

States for RL system

| State label | State |
|---|---|
| AB | state_vec(p,A,B) |
| AC | state_vec(p,A,C) |
| BC | state_vec(p,B,C) |
| BA | state_vec(p,B,A) |
| CA | state_vec(p,C,A) |
| CB | state_vec(p,C,B) |

Figure 2: On the left is a graphical depiction of a partial schedule and three candidate instructions. The table on the right shows how the RL system makes its states from this.

example, using the partial schedule and candidate instructions shown in Figure 2, scheduling instruction A, the RL system would backup values for AB, AC, and the opposite values for BA and CA.

Using this system, we performed leave-one-out cross validation across all blocks of the SPEC95 benchmark suite. Blocks with more than 800 instructions were broken into blocks of 800 or less because of memory limitations on the DEC simulator. This was true for only two applications: applu and fpppp. The RL system was trained online for 19 of the 20 applications using $\alpha = 0.05$ and an $\epsilon$-greedy exploration method with $\epsilon = 0.05$. This was repeated 20 different times, holding one program from SPEC95 out of the training each time. We then evaluated the greedy policy ($\epsilon = 0$) learned by the RL system on each program that had been held out. All ties were broken randomly. Performance was assessed the same way as before. The results for each benchmark are shown in Table 3. Overall, the RL scheduler performed only 2% slower than DEC. This is a geometric mean over all applications in the suite and on all blocks. Although the RL system did not outperform the DEC scheduler overall, it significantly outperformed DEC on the large blocks (applu-big and fpppp-big).

## 5   Conclusions

The advantages of the RL scheduler are its performance on the task, its speed, and the fact that it does not rely on any heuristics for training. Each run was much faster than with rollouts and the performance came close to the performance of the DEC scheduler. In a

| App | Ratio | App | Ratio | App | Ratio | App | Ratio |
|---|---|---|---|---|---|---|---|
| applu | 1.001 | *applu-big* | *0.959* | apsi | 1.018 | cc1 | 1.022 |
| *compress95* | *0.977* | fpppp | 1.055 | *fpppp-big* | *0.977* | go | 1.028 |
| hydro2d | 1.022 | *ijpeg* | *0.975* | li | 1.012 | m88ksim | 1.042 |
| mgrid | 1.009 | perl | 1.014 | su2cor | 1.018 | swim | 1.040 |
| tomcatv | 1.019 | turb3d | 1.218 | vortex | 1.032 | wave5 | 1.032 |

Table 3: Performance of the greedy RL-scheduler on each application in SPEC95 over all leave-one-out cross-validation runs as compared to DEC. Applications whose running time was better than DEC are shown in italics.

system where multiple architectures are being tested, RL could provide a good scheduler with minimal setup and training.

We have demonstrated two methods of instruction scheduling that do not rely on having heuristics and that perform quite well. Future work could address tying the two methods together while retaining the speed of the RL learner, issues of global instruction scheduling, scheduling loops, and validating the techniques on other architectures.

### Acknowledgments

We thank John Cavazos and Darko Stefanović for setting up the simulator and for prior work in this domain, along with Paul Utgoff, Doina Precup, Carla Brodley, and David Scheeff. We also wish to thank Andrew Barto, Andrew Fagg, and Doina Precup for comments on earlier versions of the paper. This work is supported in part by the National Physical Science Consortium, Lockheed Martin, Advanced Technology Labs, and NSF grant IRI-9503687 to Roderic A. Grupen and Andrew G. Barto. We thank various people of Digital Equipment Corporation, for the DEC scheduler and the ATOM program instrumentation tool (Srivastava & Eustace, 1994), essential to this work. We also thank Sun Microsystems and Hewlett-Packard for their support.

## References

Bertsekas, D. P. (1997). Differential training of rollout policies. In *Proc. of the 35th Allerton Conference on Communication, Control, and Computing*. Allerton Park, Ill.

Bertsekas, D. P., Tsitsiklis, J. N. & Wu, C. (1997). Rollout algorithms for combinatorial optimization. *Journal of Heuristics*.

DEC (1992). *DEC chip 21064-AA Microprocessor Hardware Reference Manual* (first edition Ed.). Maynard, MA: Digital Equipment Corporation.

Moss, J. E. B., Utgoff, P. E., Cavazos, J., Precup, D., Stefanović, D., Brodley, C. E. & Scheeff, D. T. (1997). Learning to schedule straight-line code. In *Proceedings of Advances in Neural Information Processing Systems 10 (Proceedings of NIPS'97)*. MIT Press.

Scheeff, D., Brodley, C., Moss, E., Cavazos, J. & Stefanović, D. (1997). Applying reinforcement learning to instruction scheduling within basic blocks. Technical report, University of Massachusetts, Amherst.

Sites, R. (1992). *Alpha Architecture Reference Manual*. Maynard, MA: Digital Equipment Corporation.

Srivastava, A. & Eustace, A. (1994). ATOM: A system for building customized program analysis tools. In *Proc. ACM SIGPLAN '94 Conf. on Prog. Lang. Design and Impl.* (pp. 196–205).

Stefanović, D. (1997). The character of the instruction scheduling problem. University of Massachusetts, Amherst.

Sutton, R. S. (1988). Learning to predict by the method of temporal differences. *Machine Learning*, *3*, 9–44.

Sutton, R. S. & Barto, A. G. (1998). *Reinforcement Learning. An Introduction.* Cambridge, MA: MIT Press.

Tesauro, G. & Galperin, G. R. (1996). On-line policy improvement using monte-carlo search. In *Advances in Neural Information Processing: Proceedings of the Ninth Conference*. MIT Press.